# Acoustic-Imaging Computations by Echolocating Bats: Unification of Diversely-Represented Stimulus Features into Whole Images.

**James A. Simmons**
Department of Psychology
and Section of Neurobiology,
Division of Biology and Medicine
Brown University, Providence, RI 02912.

## ABSTRACT

The echolocating bat, *Eptesicus fuscus*, perceives the distance to sonar targets from the delay of echoes and the shape of targets from the spectrum of echoes. However, shape is perceived in terms of the target's range profile. The time separation of echo components from parts of the target located at different distances is reconstructed from the echo spectrum and added to the estimate of absolute delay already derived from the arrival-time of echoes. The bat thus perceives the distance *to* targets and depth *within* targets along the same psychological range dimension, which is computed. The image corresponds to the crosscorrelation function of echoes. Fusion of physiologically distinct time- and frequency-domain representations into a final, common time-domain image illustrates the binding of within-modality features into a unified, whole image. To support the structure of images along the dimension of range, bats can perceive echo delay with a hyperacuity of 10 nanoseconds.

## THE SONAR OF BATS

Bats are flying mammals, whose lives are largely nocturnal. They have evolved the capacity to orient in darkness using a biological sonar called *echolocation*, which they use to avoid obstacles to flight and to detect, identify, and track flying insects for interception (Griffin, 1958). Echolocating bats emit brief, mostly ultrasonic sonar sounds and perceive objects from echoes that return to their ears. The bat's auditory system acts as the sonar receiver, processing echoes to reconstruct images of the objects themselves. Many bats emit frequency-modulated (FM) signals; the big brown bat, *Eptesicus fuscus*, transmits sounds with durations of several milliseconds containing frequencies from about 20 to 100 kHz arranged in two or three harmonic sweeps (Fig. 1). The images that *Eptesicus* ultimately perceives retain crucial features of the original sonar wave-

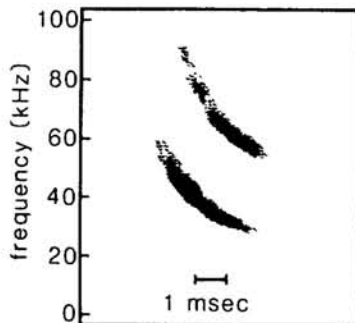

**Figure 1:** Spectrogram of a sonar sound emitted by the big brown bat, *Eptesicus fuscus* (Simmons, 1989).

forms, thus revealing how echoes are processed to reconstruct a display of the object itself. Several important general aspects of perception are embodied in specific echo-processing operations in the bat's sonar. By recognizing constraints imposed when echoes are encoded in terms of neural activity in the bat's auditory system, recent experiments have identified a novel use of time- and frequency-domain techniques as the basis for acoustic imaging in FM echolocation. The intrinsically reciprocal properties of time- and frequency-domain representations are exploited in the neural algorithms which the bat uses to unify disparate features into whole images.

## IMAGES OF SINGLE-GLINT TARGETS

A simple sonar target consists of a single reflecting point, or *glint*, located at a discrete range and reflecting a single replica of the incident sonar signal. A complex target consists of several glints at slightly different ranges. It thus reflects compound echoes composed of individual replicas of the incident sound arriving

at slightly different delays. To determine the distance to a target, or target range, echolocating bats estimate the delay of echoes (Simmons, 1989). The bat's image of a single-glint target is constructed around its estimate of echo delay, and the shape of the image can be measured behaviorally. The performance of bats trained to discriminate between echoes that jitter in delay and echoes that are stationary in delay yields a graph of the image itself (Altes, 1989), together with an indication of the accuracy of the delay estimate that underlies it (Simmons, 1979; Simmons, Ferragamo, Moss, Stevenson, & Altes, in press). Fig. 2 shows

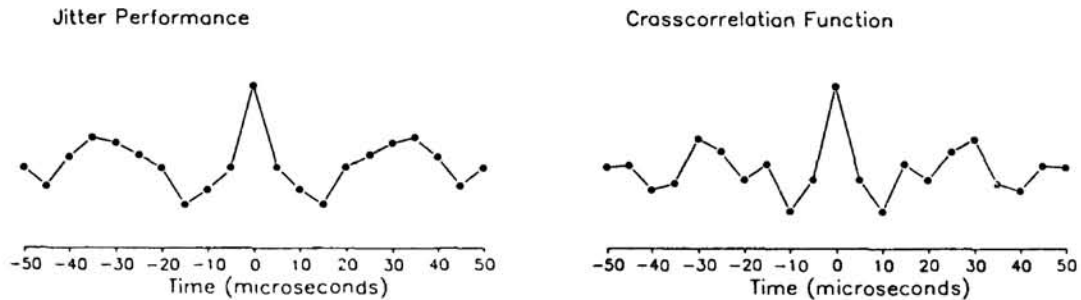

**Figure 2:** Graphs showing the bat's image of a single-glint target from jitter discrimination experiments (left) for comparison with the crosscorrelation function of echoes (right). The zero point on each time axis corresponds to the objective arrival-time of the echoes (about 3 msec in this experiment; Simmons, Ferragamo, et al., in press).

the image of a single-glint target perceived by *Eptesicus*, expressed in terms of echo delay (58 μsec/cm of range). From the bat's jitter discrimination performance, the target is perceived at its true range. Also, the image has a fine structure consisting of a central peak corresponding to the location of the target and two prominent side-peaks as ghost images located about 35 μsec or 0.6 cm nearer and farther than the main peak. This image fine structure reflects the composition of the waveform of the echoes themselves; it approximates the crosscorrelation function of echoes (Fig. 2).

The discovery that the bat perceives an image corresponding to the cross-correlation function of echoes provides a view of the hidden machinery of the bat's sonar receiver. The bat's estimate of echo delay evidently is based upon a capacity of the auditory system to represent virtually all of the information available in echo waveforms that is relevant to determining delay, including the phase of echoes relative to emissions (Simmons, Ferragamo, et al, in press). The bat's initial auditory representation of these FM signals resembles spectrograms

that consist of neural impulses marking the time-of-occurrence of successive frequencies in the FM sweeps of the sounds (Fig. 3). Each nerve im-

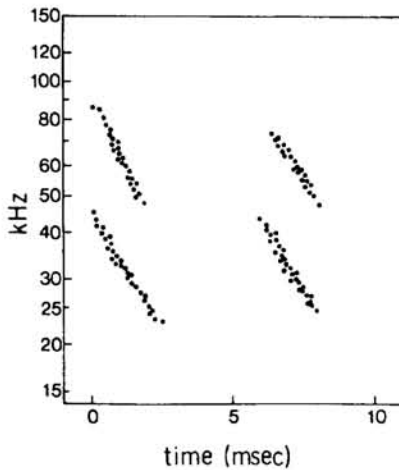

time (msec)

**Figure 3:** Neural spectrograms representing a sonar emission (left) and an echo from a target located about 1 m away (right). The individual dots are neural impulses conveying the instantaneous frequency of the FM sweeps (see Fig. 1). The 6-msec time separation of the two spectrograms indicates target range in the bat's sonar receiver (Simmons & Kick, 1984).

pulse travels in a "channel" that is tuned to a particular excitatory frequency (Bodenhamer & Pollak, 1981) as a consequence of the frequency analyzing properties of the cochlea. The cochlear filters are followed by rectification and low-pass filtering, so in a conventional sense the phase of the filtered signals is destroyed in the course of forming the spectrograms. However, Fig. 2 shows that the bat is able to reconstruct the crosscorrelation function of echoes from its spectrogram-like auditory representation. The individual neural "points" in the spectrogram signify instantaneous frequency, and the recovery of the fine structure in the image may exploit properties of instantaneous frequency when the images are assembled by integrating numerous separate delay measurements across different frequencies. The fact that the crosscorrelation function emerges from these neural computations is provocative from theoretical and technological viewpoints--the bat appears to employ novel real-time algorithms that can transform echoes into spectrograms and then into the sonar ambiguity function itself.

The range-axis image of a single-glint target has a fine structure surrounding a central peak that constitutes the bat's estimate of echo delay (Fig. 2). The width of this peak corresponds to the limiting accuracy of the bat's delay estimate, allowing for the ambiguity represented by the side-peaks located about 35 μsec away. In Fig. 2, the data-points are spaced 5 μsec apart along the time axis (approximately the Nyquist sampling interval for the bat's signals), and the true width of the central peak is poorly shown. Fig. 4 shows the performance of three *Eptesicus* in an experiment to measure this width with smaller delay steps. The

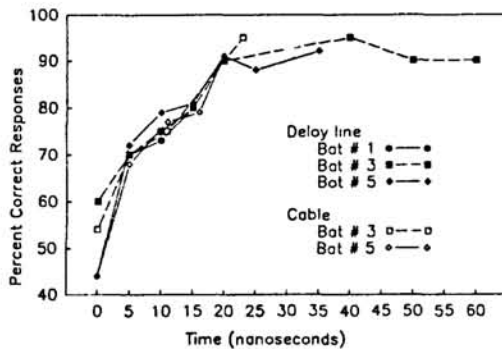

**Figure 4:** A graph of the performance of *Eptesicus* discriminating echo-delay jitters that change in small steps. The bats' limiting acuity is about 10 nsec for 75% correct responses (Simmons, Ferragamo, et al., in press).

bats can detect a shift of as little as 10 nsec as a hyperacuity (Altes, 1989) for echo delay in the jitter task. In estimating echo delay, the bat must integrate spectrogram delay estimates across separate frequencies in the FM sweeps of emissions and echoes (see Fig. 3), and it arrives at a very accurate composite estimate indeed. Timing accuracy in the nanosecond range is a previously unsuspected capability of the nervous system, and it is likely that more complex algorithms than just integration of information across frequencies lie behind this fine acuity (see below on amplitude-latency trading and perceived delay).

## IMAGES OF TWO-GLINT TARGETS

Complex targets such as airborne insects reflect echoes composed of several replicas of the incident sound separated by short intervals of time (Simmons & Chen, 1989). For insect-sized targets, with dimensions of a few centimeters, this time separation of echo components is unlikely to exceed 100 to 150 μsec. Because the bat's signals are several milliseconds long, the echoes from complex targets thus will contain echo components that largely overlap. The auditory system of *Eptesicus* has an integration-time of about 350 μsec for reception of sonar echoes (Simmons, Freedman, *et al.*, 1989). Two echo components that arrive together within this integration-time will merge together into a single compound echo having an arrival-time as a whole that indicates the delay of the first echo component, and having a series of notches in its spectrum that indicates the time separation of the first and second components. In the bat's auditory representation, echo delay corresponds to the time separation of the emission and echo spectrograms (see Fig. 3), while the notches in the compound echo spectrum appear as "holes" in the spectrogram--that is, as frequencies that fail to appear in echoes. The location and spacing of these notches or holes in *frequency* is related to the separation of the two echo components in *time*. The crucial point is that the constraint imposed by the 350-μsec integration-time for echo reception disperses the information required to reconstruct the detailed range

structure of the complex target into both the time and the frequency dimensions of the neural spectrograms.

*Eptesicus* extracts an estimate of the overall delay of the waveform of compound echoes from two-glint targets. This time estimate leads to a range-axis image of the closer of the two glints in the target (the target's leading edge). This part of the image exhibits the same properties as the image of a single-glint target--it is encoded by the time-of-occurrence of neural discharges in the spectrograms and it resembles the crosscorrelation function for the first echo component (Simmons, Moss, & Ferragamo, 1990; Simmons, Ferragamo, et al., in press; see Simmons, 1989). The bat also perceives a range-axis image of the farther of the two glints (the target's trailing edge). This image is located at a perceived distance that corresponds to the bat's estimate of the time separation of the two echo components that make up the compound echo. Fig. 5 shows the performance of *Eptesicus* in a jitter discrimination experiment in which one of the

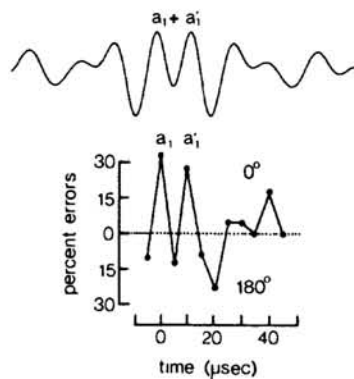

Figure 5: A graph comparing the crosscorrelation function of echoes from a two-glint target with a delay separation of 10 μsec (top) with the bat's jitter discrimination performance using this compound echo as a stimulus (bottom). The two glints are indicated as $a_1$ and $a_1'$ (Simmons, 1989).

jittering stimulus echoes contained two replicas of the bat's emitted sound separated by 10 μsec. The bat perceives two distinct reflecting points along the range axis. Both glints appear as events along the range axis in a time-domain image even though the existence of the second glint could only be inferred from the frequency domain because the delay separation of 10 μsec is much shorter than the receiver's integration time. The image of the second glint resembles the crosscorrelation function of the later of the two echo components. The bat adds it to the crosscorrelation function for the earlier component when the whole image is formed.

## ACOUSTIC-IMAGE PROCESSING BY FM BATS

Somehow *Eptesicus* recovers sufficient information from the timing of neural discharges across the frequencies in the FM sweeps of emissions and echoes to reconstruct the crosscorrelation function of echoes from the first glint in the complex target and to estimate delay with nanosecond accuracy. This fundamentally time-domain image is derived from the processing of information initially also represented in the time domain, as demonstrated by the occurrence of changes in apparent delay as echo amplitude increases or decreases: The location of the perceived crosscorrelation function for the first glint can be shifted by predictable amounts along the time axis according to the separately-measured amplitude-latency trading relation for *Eptesicus* (about -17 µsec/dB; Simmons, Moss, & Ferragamo, 1990; Simmons, Ferragamo, et al., in press), indicating that neural response latency--that is, neural discharge timing--conveys the crucial information about delay in the bat's auditory system.

The second glint in the complex target manifests itself as a crosscorrelation-like image component, too. However, the bat must transform spectral information into the time domain to arrive at such a time- or range-axis representation for the second glint. This transformed time-domain image is added to the time-domain image for the first glint in such a way that the absolute range of the second glint is referred to that of the first glint. Shifts in the apparent range of the first glint caused by neural discharges undergoing amplitude-latency trading will carry the image of the second glint along with it to a new range value (Simmons, Moss, & Ferragamo, 1990). Evidently, the psychological dimension of absolute range supports the image of the target as a whole. This helps to explain the bat's extraordinary 10-nsec accuracy for perceiving delay. For the psychological range or delay axis to accept fine-grain range information about the separation of glints in complex targets, its intrinsic accuracy must be adequate to receive the information that is transformed from the frequency domain. The bat achieves fusion of image components by transforming one component into the numerical format for the other and then adding them together. The experimental dissociation of the images of the first and second glints from different effects of latency shifts demonstrates the independence of their initial physiological representations. Furthermore, the expected latency shift does not occur for frequencies whose amplitudes are low because they coincide with spectral notches; the bat's fine nanosecond acuity thus seems to involve removal of discharges at "untrustworthy" frequencies prior to integration of discharge timing across frequencies. The delay-tuning of neurons is usually thought to represent the conversion of a temporal code (timing of neural discharges) into a "place" code (the location of activity on the neural map). The bat's unusual acuity of 10 nsec suggests that this conversion of a temporal to a "place" code is only partial.

Not only does the site of activity on the neural map convey information about delay, but the timing of discharges in map neurons may also play a critical role in the map-reading operation. The bat's fine acuity may emerge in the behavioral data because initial neural encoding of the stimulus conditions in the jitter task involves the same parameter of neural responses--timing--that later is intimately associated with map-reading in the brain. Echolocation may thus fortuitously be a good system in which to explore this basic perceptual process.

## Acknowledgments

Research supported by grants from ONR, NIH, NIMH, DRF, and SDF.

## References

R. A. Altes (1989) Ubiquity of hyperacuity, *J. Acoust. Soc. Am.* 85: 943-952.

R. D. Bodenhamer & G. D. Pollak (1981) Time and frequency domain processing in the inferior colliculus of echolocating bats, *Hearing Res.* 5: 317-355.

D. R. Griffin (1958) *Listening in the Dark*, Yale Univ. Press.

J. A. Simmons (1979) Perception of echo phase information in bat sonar, *Science*, 207: 1336-1338.

J. A. Simmons (1989) A view of the world through the bat's ear: the formation of acoustic images in echolocation, *Cognition* 33: 155-199.

J. A. Simmons & L. Chen (1989) The acoustic basis for target discrimination by FM echolocating bats, *J. Acoust. Soc. Am.* 86: 1333-1350.

J. A. Simmons, M. Ferragamo, C. F. Moss, S. B. Stevenson, & R. A. Altes (in press) Discrimination of jittered sonar echoes by the echolocating bat, *Eptesicus fuscus*: the shape of target images in echolocation, *J. Comp. Physiol. A*.

J. A. Simmons, E. G. Freedman, S. B. Stevenson, L. Chen, & T. J. Wohlgenant (1989) Clutter interference and the integration time of echoes in the echolocating bat, *Eptesicus fuscus*, J. Acoust. Soc. Am. 86: 1318-1332.

J. A. Simmons & S. A. Kick (1984) Physiological mechanisms for spatial filtering and image enhancement in the sonar of bats, *Ann. Rev. Physiol.* 46: 599-614.

J. A. Simmons, C. F. Moss, & M. Ferragamo (1990) Convergence of temporal and spectral information into acoustic images perceived by the echolocating bat, *Eptesicus fuscus*, *J. Comp. Physiol. A* 166: